# Learning Time-Intensity Profiles of Human Activity using Non-Parametric Bayesian Models

**Alexander T. Ihler**   **Padhraic Smyth**
Donald Bren School of Information and Computer Science
U.C. Irvine
*ihler@ics.uci.edu*     *smyth@ics.uci.edu*

## Abstract

Data sets that characterize human activity over time through collections of time-stamped events or counts are of increasing interest in application areas as human-computer interaction, video surveillance, and Web data analysis. We propose a non-parametric Bayesian framework for modeling collections of such data. In particular, we use a Dirichlet process framework for learning a set of intensity functions corresponding to different categories, which form a basis set for representing individual time-periods (e.g., several days) depending on which categories the time-periods are assigned to. This allows the model to learn in a data-driven fashion what "factors" are generating the observations on a particular day, including (for example) weekday versus weekend effects or day-specific effects corresponding to unique (single-day) occurrences of unusual behavior, sharing information where appropriate to obtain improved estimates of the behavior associated with each category. Applications to real–world data sets of count data involving both vehicles and people are used to illustrate the technique.

## 1 Introduction

As sensor and storage technologies continue to improve in terms of both cost and performance, increasingly rich data sets are becoming available that characterize the rhythms of human activity over time. Examples include logs of radio frequency identification (RFID) tags, freeway traffic over time (loop-sensor data), crime statistics, email and Web access logs, and many more. Such data can be used to support a variety of different applications, such as classification of human or animal activities, detection of unusual events, or to support the broad understanding of behavior in a particular context such as the temporal patterns of Web usage.

To ground the discussion, consider data consisting of a collection of individual or aggregated events from a single sensor, e.g., a time-stamped log recording every entry and exit from a building, or the timing and number of highway traffic accidents. For example, Figure 1 shows several days worth of data from a building log, smoothed so that the similarities in patterns are more readily visible.

Of interest is the modeling of the underlying intensity of the process generating the data, where intensity here refers to the rate at which events occur. These processes are typically inhomogeneous in time (as in Figure 1), as they arise from the aggregated behavior of individuals, and thus exhibit a temporal dependence linked to the rhythms of the underlying human activity. The complexity of this temporal dependence is application-dependent and generally unknown before observing the data, suggesting that non- or semi-parametric methods (methods whose complexity is capable of growing as the number of observations increase) may be particularly appropriate.

Formulating the underlying event generation as an inhomogeneous Poisson process is a common first step (see, e.g., [1, 4]), as it allows the application of various classic density estimation techniques to estimate the time-dependent intensity function (a normalized version of the rate function; see Sec-

tion 2). Techniques used in this context include kernel density estimation [2], wavelet analysis [3], discretization [1], and nonparametric Bayesian models [4, 5].

Among these, nonparametric Bayesian approaches have a number of appealing advantages. First, they allow us to represent and reason about uncertainty in the intensity function, providing not just a single estimate but a distribution over functions. Second, the Bayesian framework provides natural methods for model selection, allowing the data to be naturally explained by a parsimonious set of intensity functions, rather than using the most complex explanation (though similar effects may be achieved using penalized likelihood functions [3]). Finally, Bayesian methods generalize to multiple or hierarchical models, which allow information to be shared among several related but differing sets of observations (e.g., multiple days of data). This second point is crucial for many problems, as we rarely obtain many observations of exactly the same process under exactly the same conditions; instead, we observe multiple instances which are thought to be similar, but may in fact represent any number of slightly differing circumstances. For example, behavior may be dependent on not only time of day but also day of week, type of day (weekend or weekday), unobserved factors such as the weather, or other unusual circumstances. Sharing information allows us to improve our model, but we should only do so where appropriate (itself best indicated by similarity in the data). By being Bayesian, we can remain agnostic of what data should be shared and reason over our uncertainty in this structure.

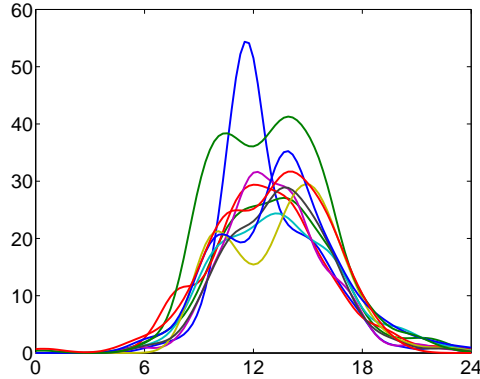

Figure 1: Count data from a building entry log observed on ten Mondays, each smoothed using a kernel function [2, 6] to enable visual comparison.

In what follows we propose a non-parametric Bayesian framework for modeling intensity functions for event data over time. In particular, we describe a Dirichlet process framework for learning the unknown rate functions, and learn a set of such functions corresponding to different categories. Individual time-periods (e.g., individual days) are then represented as additive combinations of intensity functions, depending on which categories are assigned to each time-period. This allows the model to learn in a data-driven fashion what "factors" are generating the observations on a particular day, including (for example) weekday versus weekend effects as well as day-specific effects corresponding to unusual behavior present only on a single day. Applications to two real–world data sets, a building access log and accident statistics, are used to illustrate the technique.

We will discuss in more detail in the sections that follow how our proposed approach is related to prior work on similar topics. Broadly speaking, from the viewpoint of modeling of inhomogeneous time-series of counts, our work extends the work of [4] to allow sharing of information among multiple, related processes (e.g., different days). Our approach can also be viewed as an alternative to the hierarchical Dirichlet process (HDP, [7]) for problems where the patterns across different groups are much more constrained than would be expected under an HDP model.

## 2 Poisson processes

A common model for continuous-time event (counting) data is the Poisson process [8]. As the discrete Poisson distribution is characterized by a rate parameter $\lambda$, the Poisson process[1] is characterized by a rate function $\lambda(t)$; it has the property that over any given time interval $\mathcal{T}$, the number of events occurring within that time is Poisson with rate given by $\lambda = \int_{\mathcal{T}} \lambda(t)$. We shall use a Bayesian semi-parametric model for $\lambda(t)$, described next.

Let us suppose that we have a single collection of event times $\{\tau_i\}$ arising from a Poisson process with rate function $\lambda(t)$, i.e.,

$$\{\tau_i\} \sim \mathrm{P}[\tau \, ; \, \lambda(t)] \tag{1}$$

where $\lambda(t)$ is defined on $t \in [-\infty, \infty]$. We may write $\lambda(t) = \gamma f(t)$, where $\gamma = \int_{-\infty}^{\infty} \lambda(t)$ and $f(t)$ is the *intensity function*, a normalized version of the rate function. A Bayesian model places prior distributions on these quantities; by selecting a parametric prior for $\gamma$ and a nonparametric prior for $f(t)$, we obtain a semi-parametric prior for $\lambda(t)$. Specifically, we choose

$$\gamma \sim \Gamma(a,b) \qquad f(t) = \int K(t;\theta) dG(\theta) \qquad G \sim \mathrm{DP}[\alpha G_0]$$

where $\Gamma$ is the gamma distribution, $K$ is a kernel function (for example a Gaussian distribution) and DP is a Dirichlet process [9] with parameter $\alpha$ and base distribution $G_0$. The Dirichlet process provides a nonparametric prior for $f(t)$, such that (with probability one) $f$ has the form of a mixture model with infinitely many components: $f(t) = \sum_j w_j K(t;\theta_j)$. If desired we may also place prior distributions on some or all of these quantities (e.g., $\alpha$, $\{a,b\}$, or the parameters of $G_0$) as well.

Dirichlet processes and their variations [7, 9–11] have gained recent attention for their ability to provide representations consisting of arbitrarily large mixture models. In particular, they have been the subject of recent work in modeling intensity functions for Poisson processes defined over time [4] and space–time [5].

## 2.1 Monte Carlo Inference

For the Poisson process model just described, the likelihood of the data $\{\tau_i\}$, $i = 1 \ldots N$ at some time $T$ is given by

$$p(\{\tau_i\}; \gamma, f(\cdot)) = \exp\left(-\int_{-\infty}^{T} \gamma f(t)\right) \gamma^N \prod_i f(\tau_i)$$

which, as $T \to \infty$ (i.e., as we observe a complete data set) becomes

$$p(\{\tau_i\}; \gamma, f(\cdot)) = \left[\exp(-\gamma)\gamma^N\right] \left[\prod_i f(\tau_i)\right] \qquad (2)$$

The rightmost term (term involving $f$) has the same form as the likelihood of the $\tau_i$ as i.i.d. samples from the mixture model distribution defined by $f$. As in many mixture model applications, it will be helpful to create auxiliary assignment variables $z_i$ for each $\tau_i$, indicating with which of the mixture components the sample $\tau_i$ is associated. The complete data likelihood is then

$$p(\{\tau_i, z_i\}; \gamma, f(\cdot)) = \left[\exp(-\gamma)\gamma^N\right] \left[\prod_i w_{z_i} K(\tau_i; \theta_{z_i})\right].$$

Inference is typically accomplished using Markov chain Monte Carlo (MCMC) sampling [9]. Specifically, although the posterior for $\gamma$ has a simple closed form, $p(\gamma | \{\tau_i\}) \propto \Gamma(N + a, 1 + b)$, sampling from $f$ is more complicated. Samples from $f$ can be drawn in a variety of ways. One of the most common methods is the so-called "Chinese Restaurant Process" (CRP, [7, 9]), in which the relative weights $w_j$ are marginalized out while drawing the assignment variables $z_i$. Such exact sampling approaches work by exploiting the fact that only a finite number of the mixture components are occupied by the data; by treating the unoccupied clusters as a single group, the infinite number of potential associations can be treated as a finite number. The operations involved (such as sampling values for $\theta_j$ given a collection of associated event times $\tau_i$) are easier for certain choices of $K$ and $G$ than others; for example using a Gaussian kernel and normal-Wishart distribution, the necessary quantities have convenient closed forms [9].

Another, more brute-force way around the issue of having infinitely many mixture components is to perform *approximate* sampling using a "truncated" Dirichlet process representation [12, 13]. As described in [12], for a given $\alpha$, data set size $N$, and tolerance $\epsilon$, one can compute a maximum number of components $M$ necessary to approximate the Dirichlet process with a Dirichlet distribution using the relation

$$\epsilon \approx 4 N \exp[-(M-1)/\alpha]$$

and in this manner, can work with finite numbers of mixture components. This representation will prove useful in Section 3.

The truncated DP approximation is helpful primarily because it allows us to sample the (complete) function $f(t)$ (as compared to only the "occupied" part in the CRP formulation). Given a set of assignments $\{z_i\}$ occupying (arbitrarily numbered) clusters $1 \ldots J$, we can sample the weights $w_j$ in two steps. First, we sample the occupied mixture weights, $w_j$ ($j \leq J$), and the total unoccupied weight $\bar{w} = \sum_{J+1}^{\infty} w_j$, by drawing independent, Gamma-distributed random variables according to $\Gamma(N_j, 1)$ and $\Gamma(\alpha, 1)$, respectively, and normalizing them to sum to one. The values of weights $w_j$ in the unoccupied clusters ($j > J$) can then be sampled given $\bar{w}$ using the stick–breaking representation of Sethuraman [14].

Note that the truncated DP approximation highlights the importance of also sampling $\alpha$ if we hope for our representation to act non-parametric in the sense that it may grow more complex as the data increase, since for a fixed $\alpha$ and $\epsilon$ the number of components $M$ is quite insensitive to $N$. For more details on sampling such hyper-parameters see e.g. [10].

### 2.2   Finite Time Domains

Our description of non-parametric Bayesian techniques for Poisson processes has so far made implicit use of the fact that the domain of $f(t)$ is infinite. When the domain of $f$ is finite, for example $[0, 1]$, a few minor complications arise. For example, the kernel functions $K(\cdot)$ should properly be defined as positive only on this interval. One possible solution to this issue is to use an alternate kernel function, such as the Beta distribution [4]. However, this means that posterior sampling of the parameters $\theta$ is no longer possible in closed form. Although methods such as Metropolis-Hastings may be used [4], they can be highly dependent on the choice of proposal density.

Here, we take a slightly different approach, drawing truncated Gaussian kernels with parameters sampled from a truncated Normal-Wishart distribution. Specifically, we define

$$K(t; \theta = [\mu, \sigma^2]) = \frac{\mathcal{N}(t; \mu, \sigma^2)\chi_1(\mu)}{\int_0^1 \mathcal{N}(x; \mu, \sigma^2)dx} \qquad [\mu, \sigma^2] \sim \chi_1(\mu)\,\chi_1(\sigma)\,\mathcal{NW}(\mu, \sigma^2)$$

where $\chi_1(t)$ is one on $[0, 1]$ and zero elsewhere and $\mathcal{NW}$ is the normal-Wishart distribution. Sampling in this model turns out to be relatively simple and efficient using rejection methods. Given the restrictions imposed on $\mu$ and $\sigma$, one can show that the normalizing quantity $Z = \int_0^1 \mathcal{N}(x; \mu, \sigma^2)$ is always greater than one-third. Thus, to sample from the posterior we simply draw from the original, closed form posterior distribution, discarding (and re-sampling) if $\mu \notin [0, 1]$, $\sigma \notin [0, 1]$, or with probability $1 - (3Z)^{-1}$.

## 3   Categorical Models

As mentioned in the introduction, we often have *several* collections $d = 1 \ldots D$ of observations, $\{\tau_{di}\}$ with $i = 1 \ldots N_d$, arising from $D$ instances of the same or similar processes. If these processes are known to be identical and independent, sharing information among them is relatively easy—we obtain $D$ observations $N_d$ with which to estimate $\gamma$, and the $\tau_{di}$ are collectively used to estimate $f(t)$. However, if these processes are not necessarily identical, sharing information becomes more difficult.

Yet it is just this situation which is most typical. Again consider Figure 1, which shows event data from ten different Mondays. Clearly, there is a great deal of consistency in both size and shape, although not every day is exactly the same, and one or two stand out as different. Were we to also look at, for example, Sundays or Tuesdays (as we do in Section 4), we would see that although Sunday and Monday appear quite different and, one suspects, have little shared information, Monday and Tuesday appear relatively similar and this similarity can probably be used to improve our rate estimates for both days.

In this example, we might reasonably assume that the category memberships are known (for example, whether a given day is a weekday or weekend, or a Monday or Tuesday), though we shall relax this assumption in later sections. Then, given a structure of potential relationships, what is a reasonable model for sharing information among categories? There are, of course, many possible choices; we use a simple additive model, described in the next section.

## 3.1 Additive Models

The intuition behind an additive model is that the data arises from the superposition of several underlying causes present during the period of interest. Again, we initially assume that the category memberships are known; thus, if a category is associated with a particular day, the activity profile associated with that category will be observed, along with additional activity arising from each of the other categories present.

Let us associate a rate function $\lambda_c(t) = \gamma_c f_c(t)$ with each category in our model. We define the rate function of a given day $d$ to be the sum of the rate functions of each category to which $d$ belongs. Denoting by $s_{dc}$ the (binary-valued) membership indicator, i.e., that category $c$ is present during day $d$, we have that $\lambda_d(t) = \sum_{c:s_{dc}=1} \lambda_c(t)$.

At first, this model might seem quite restrictive. However, it matches our intuition of how the data is generated, stemming from the presence or absence of a particular behavioral pattern associated with some underlying cause (such as it being a work day). In fact, we do not want a model which is too flexible, such as a linear combination of patterns, since it is not physically meaningful to say, for example, that a day is only "part" Monday. To learn the profiles associated with a given cause (e.g., things that happen every day versus only on weekdays or only on Mondays), it makes sense to take an "all or nothing" model where the pattern is either present, or not. This also suggests that other methods of coupling Dirichlet processes, such as the hierarchical Dirichlet process [7], may be too flexible. The HDP couples the parameters of components across levels, but only loosely relates the actual shape of the profile, since it allows components to be larger or smaller (or even disappear completely). In [7], this is a desirable quality, but in our application it is not. Using an additive model allows both a consistent size and shape to emerge for each category, while associating deviations from that profile to categories further down in the hierarchy.

Inference in this system is not significantly more difficult than in the single rate function case (Section 2). We define the association as $[y_{di}, z_{di}]$, where $y_{di}$ indicates which of the categories generated event $\tau_{di}$. It is easy to sample $y_{di}$ according to $p(y_{di} = c|\{\lambda_c(t)\}) \propto [\lambda_c(\tau_{di})] / [\sum_{c'} \lambda_{c'}(\tau_{di})]$.

## 3.2 Sampling Membership

Of course, it is frequently the case that the membership(s) of each collection of data are not known precisely. In an extreme case, we may have no idea which collections are similar and should be grouped together and wish to find profiles in an unsupervised manner. More commonly, however, we have some prior knowledge and interpretation of the profiles but do not wish to strictly enforce a known membership. For example, if we create categories with assigned meanings (weekdays, weekends, Sundays, Mondays, and so on), a day which is nominally a Monday but also happens to be a holiday, closure, or other unusual circumstances may be completely different from other Monday profiles. Similarly, a day with unusual extra activity (receptions, talks, etc.) may see behavior unique to its particular circumstances and warrant an additional category to represent it.

We can accommodate both these possibilities by also sampling the values of the membership indicator variables $s_{dc}$, i.e., the binary indicator that day $d$ sees behavior from category $c$. To this end, let us assume we have some prior knowledge of these membership probabilities, $p_{dc}(s_{dc})$; we may then re-sample from their posterior distributions at each iteration of MCMC.

This sampling step is difficult to do outside the truncated representation. Although up until this point we could easily have elected to use, for example, the CRP formulation for sampling, the association variables $\{y_{di}, z_{di}\}$ are tightly coupled with the memberships $s_{dc}$ since if any $y_{di} = c$ we must have that $s_{dc} = 1$. Instead, to sample the $s_{dc}$ we condition on the truncated rate functions $\lambda_c(t)$, with truncation depth $M$ chosen to provide arbitrarily high precision. The likelihood of the data under these rate functions for any values of $\{s_{dc}\}$ can then be computed directly via (2) where

$$\gamma = \sum_c s_{dc}\gamma_c \qquad \text{and} \qquad f(t) = \gamma^{-1} \sum_c s_{dc}\lambda_c(t).$$

In practice, we propose changing the value of each membership variable $s_{dc}$ individually given the others, though more complex moves could also be applied. This gives the following sequence of MCMC sampling: (1) given a truncated representation of the $\{\lambda_c(t)\}$, sample membership variables $\{s_{dc}\}$; (2) given $\{\lambda_c(t)\}$ and $\{s_{dc}\}$, sample associations $\{z_{di}\}$; (3) given associations $\{z_{di}\}$, sample

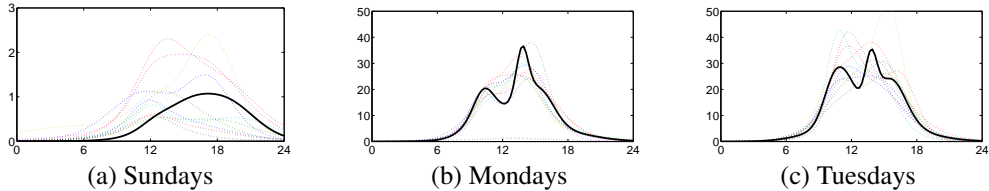

(a) Sundays          (b) Mondays          (c) Tuesdays

Figure 2: Posterior mean estimates of rate functions for building entry log data, estimated individually for each day (dotted) and learned by sharing information among multiple days (solid) for (a) Sundays, (b) Mondays, and (c) Tuesdays. Sharing information among similar days gives greatly improved estimates of the rate functions, resolving otherwise obscured features such as the decrease during and increase subsequent to lunchtime.

category magnitudes $\{\gamma_c\}$ and a truncated representation of each $f_c(t)$ consisting of weights $\{w_j\}$ and parameters $\{\theta_j\}$.

# 4 Experiments

In this section we consider the application of our model to two data sets, one (mentioned previously) from the entry log of people entering a large campus building (produced by optical sensors at the front door), and the other from a log of vehicular traffic accidents. By design, both data sets contain about ten weeks worth of observations. In both cases, we have a plausible prior structure for and interpretation of the categories, i.e., that similar days will have similar profiles. To this end, we create categories for "all days", "weekends", "weekdays", and "Sundays" through "Saturdays". Each of these categories has a high probability ($p_{dc} = .99$) of membership for each eligible day. To account for the possibility of unusual increases in activity, we also add categories unique to each day, with lower prior probability ($p_{dc} = .20$) of membership. This allows but discourages each day to add a new category if there is evidence of unusual activity.

## 4.1 Building Entry Data

To see the improvement in the estimated rate functions when information is shared among similar days, Figure 2 shows results from three different days of the week (Sunday, Monday, Tuesday). Each panel shows the estimated profiles of each of the ten days estimated individually (using only that day's observations) under a Dirichlet process mixture model (dotted lines). Superimposed in each panel is a single, black curve corresponding to the total profile for that day of week estimated using our categorical model; so, (a) shows the sum of the rate functions for "all days", "weekends", and "Sundays", while (b) shows the sum of "all days", "weekdays", and "Mondays". We use the same prior distributions for both the individual estimates and the shared estimate.

Several features are worth noting. First, by sharing several days worth of observations, the model can produce a much more accurate estimate of the profiles. In this case, no single day contains enough observations to be confident about the details of the rate function, so each individually–estimated rate function appears relatively smooth. However, when information from other days is included, the rate function begins to resolve into a clearly bi-modal shape for weekdays. This "bi-modal" rate behavior is quite real, and corresponds to the arrival of occupants in the morning (first mode), a lull during lunchtime, and a larger, narrower second peak as most occupants return from lunch.

Second, although Monday and Tuesday profiles appear similar, they also have distinct behavior, such as increased activity late Tuesday morning. This behavior too has some basis in reality, corresponding to a regular weekly meeting held around lunchtime over most (though not quite all) of the weeks in question. The breakdown of a particular day (the first Tuesday) into its component categories is shown in Figure 3. As we might expect, there is little consistency between weekdays and weekends, quite a bit of similarity among weekdays and among just Tuesdays, and (for this particular day) very little to set it apart from other Tuesdays.

We can also check to see that the category memberships $s_{dc}$ are being used effectively. One of the Mondays in our data set fell on a holiday (the individual profile very near zero). If we average the probabilities computed during MCMC to estimate the posterior probability of the $s_{dc}$ for that

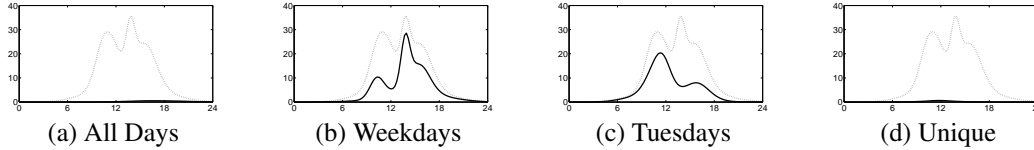

(a) All Days      (b) Weekdays      (c) Tuesdays      (d) Unique

Figure 3: Posterior mean estimates of the rate functions for each category to which the first Tuesday data might belong. For comparison, the total rate (sum of all categories) is shown as the dotted line. (a) The "all days" category is small, indicating little consistency in the data between weekdays and weekends; (b) the "weekdays" category is larger, and contains a component which appears to correspond to the occupants' return from lunch; (c) the "Tuesday" category has modes in the morning and afternoon, perhaps capturing regular meetings or classes; (d) the "unique" category (a category unique to this particular day) shows little or no activity.

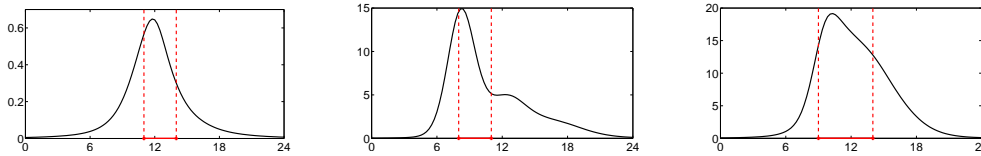

Figure 4: Profiles associated with individual-day categories in the entry log data for several days with known events (periods between dashed vertical lines). The model successfully learns which days have significant unusual activity and associates reasonable profiles with that activity (note that increases in entrance count rate typically occurs shortly before or at the beginning of the event time).

particular day, we find that it has near-zero probability of belonging to either the weekday or Monday categories, and uses only the all-day and unique categories.

We can also examine days which have high probability of requiring their own category (indicating unusual activity). For this data set, we also have partial ground truth, consisting of a number of dates and times when activities were scheduled to take place in the building. Figure 4 shows three such days, and the corresponding rate profiles associated with their single-day categories. Again, all three days are estimated to have additional activity, and the period of time for that activity corresponds well with the actual start and end time shown in the schedule (dashed vertical lines).

## 4.2 Vehicular Accident Data

Our second data set consists of a database of vehicular accident times recorded by North Carolina police departments. As we might expect of driving patterns, there is still less activity on weekends, but far more than was observed in the campus building log.

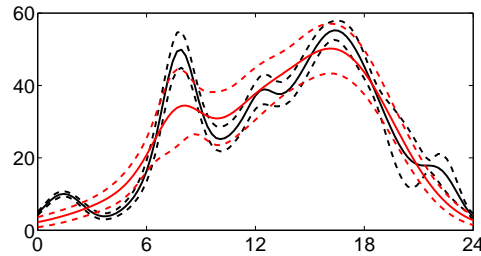

Figure 5: Posterior mean and uncertainty for a single day of accident data, estimated individually (red) and with data sharing (black). Sharing data considerably reduces the posterior uncertainty in the profile shape.

As before, sharing information allows us to decrease our posterior uncertainty on the rate for any particular day. Figure 5 quantifies this idea by showing the posterior means and (pointwise) two-sigma confidence intervals for the rate function estimated for the same day (the first Monday in the data set) using that day's data only (red curves) and using the category-based additive model (black). The additive model leverages the additional data to produce much tighter estimates of the rate profile.

As with the previous example, the additional data also helps resolve detailed features of each day's profile, as seen in Figure 6. For example, the weekday profiles show a tri-modal shape, with one mode corresponding to the morning commute, a small mode around noon, and another large mode

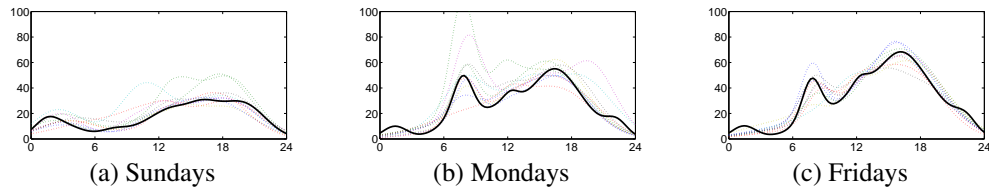

| (a) Sundays | (b) Mondays | (c) Fridays |

Figure 6: Posterior mean estimates of rate functions for vehicular accidents, estimated individually for each day (dotted) and with sharing among multiple days (solid) for (a) Sundays, (b) Mondays, and (c) Fridays. As in Figure 2, sharing information helps resolve features which the individual days do not have enough data to reliably estimate.

around the evening commute. This also helps make the pattern of deviation on Friday clear, showing (as we would expect) increased activity at night.

## 5   Conclusions

The increasing availability of logs of "human activity" data provides interesting opportunities for the application of statistical learning techniques. In this paper we proposed a non-parametric Bayesian approach to learning time-intensity profiles for such activity data, based on an inhomogeneous Poisson process framework. The proposed approach allows collections of observations (e.g., days) to be grouped together by category (day of week, weekday/weekend, etc.) which in turn leverages data across different collections to yield higher quality profile estimates. When the categorization of days is not a priori certain (e.g., days that fall on a holiday or days with unusual non-recurring additional activity) the model can infer the appropriate categorization, allowing (for example) automated detection of unusual events. On two large real-world data sets the model was able to infer interpretable activity profiles that correspond to real-world phenomena. Directions for further work in this area include richer models that allow for incorporation of observed covariates such as weather and other exogenous phenomena, as well as modeling of multiple spatially-correlated sensors (e.g., loop sensor data for freeway traffic).

## Footnotes

[1]Here, we shall use the term Poisson process interchangeably with *inhomogeneous* Poisson process, meaning that the rate is a non-constant function of time $t$.

## References

[1]  S. Scott and P. Smyth. The Markov modulated Poisson process and Markov Poisson cascade with applications to web traffic data. *Bayesian Statistics*, 7:671–680, 2003.

[2]  R. Helmers, I.W. Mangku, and R. Zitikis. Consistent estimation of the intensity function of a cyclic Poisson process. *J. Multivar. Anal.*, 84(1):19–39, January 2003.

[3]  R. Willett and R. Nowak. Multiscale Poisson intensity and density estimation. *submitted to IEEE Trans. IT*, January 2005.

[4]  A. Kottas. Bayesian nonparametric mixture modeling for the intensity function of non-homogeneous Poisson processes. Technical Report ams2005-02, Department of Applied Math and Statistics, U.C. Santa Cruz, Santa Cruz, CA, 2005.

[5]  A. Kottas and B. Sanso. Bayesian mixture modeling for spatial Poisson process intensities, with applications to extreme value analysis. Technical Report ams2005-19, Dept. of Applied Math and Statistics, U.C. Santa Cruz, Santa Cruz, CA, 2005.

[6]  B.W. Silverman. *Density Estimation for Statistics and Data Analysis*. Chapman & Hall, NY, 1986.

[7]  Y.W. Teh, M.I. Jordan, M.J. Beal, and D.M. Blei. Hierarchical Dirichlet processes. In *NIPS 17*, 2004.

[8]  D.R. Cox. Some statistical methods connected with series of events. *J. R. Stat. Soc. B*, 17:129–164, 1955.

[9]  R.M. Neal. Markov chain sampling methods for Dirichlet process mixture models. *J. of Comp. Graph. Stat.*, 9:283–297, 2000.

[10]  M.D. Escobar and M. West. Bayesian density estimation and inference using mixtures. *J. Amer. Stat. Assoc.*, 90:577–588, 1995.

[11]  L.F. James. Functionals of Dirichlet processes, the Cifarelli-Reganzzini identity and Beta-Gamma processes. *Ann. Stat.*, 33(2):647–660, 2005.

[12]  H. Ishwaran and L.F. James. Gibbs sampling methods for stick-breaking priors. *J. Amer. Stat. Assoc.*, 96:161–173, 2001.

[13]  H. Ishwaran and L.F. James. Approximate Dirichlet process computing in finite normal mixtures: smoothing and prior information. *J. Comp. Graph. Statist.*, 11:508–532, 2002.

[14]  J. Sethuraman. A constructive definition of Dirichlet priors. *Statistica Sinica*, 4:639–650, 1994.
